# Planar Hidden Markov Modeling:
# from Speech to Optical Character Recognition

**Esther Levin and Roberto Pieraccini**
ATT Bell Laboratories
600 Mountain Ave.
Murray Hill, NJ 07974

## Abstract

We propose in this paper a statistical model (planar hidden Markov model - PHMM) describing statistical properties of images. The model generalizes the single-dimensional HMM, used for speech processing, to the planar case. For this model to be useful an efficient segmentation algorithm, similar to the Viterbi algorithm for HMM, must exist. We present conditions in terms of the PHMM parameters that are sufficient to guarantee that the planar segmentation problem can be solved in polynomial time, and describe an algorithm for that. This algorithm aligns optimally the image with the model, and therefore is insensitive to elastic distortions of images. Using this algorithm a joint optimal segmentation and recognition of the image can be performed, thus overcoming the weakness of traditional OCR systems where segmentation is performed independently before the recognition leading to unrecoverable recognition errors.

The PHMM approach was evaluated using a set of isolated hand-written digits. An overall digit recognition accuracy of 95% was achieved. An analysis of the results showed that even in the simple case of recognition of isolated characters, the elimination of elastic distortions enhances the performance significantly. We expect that the advantage of this approach will be even more significant for tasks such as connected writing recognition/spotting, for which there is no known high accuracy method of recognition.

## 1 Introduction

The performance of traditional OCR systems deteriorate very quickly when documents are degraded by noise, blur, and other forms of distortion. The main reason for such deterioration is that in addition to the intra-class character variability caused by distortion, the segmentation of the text into words and characters becomes a nontrivial task. In most of the traditional systems, such segmentation is done *before* recognition, leading to many recognition errors, since recognition algorithms cannot usually recover from errors introduced in the segmentation phase. Moreover, in many cases the segmentation is ill-defined, since many plausible segmentations might exist, and only grammatical and linguistic analysis can find the "right " one. To address these problems, an algorithm is needed that can :

- be tolerant to distortions leading to intra-class variability

- perform segmentation together with recognition, thus jointly optimizing both processes, while incorporating grammatical/linguistic constraints.

In this paper we describe a planar segmentation algorithm that has the above properties. It results from a direct extension of the Viterbi (Forney, 1973) algorithm, widely used in automatic speech recognition, to two-dimensional signals.

In the next section we describe the basic hidden Markov model and define the segmentation problem. In section 3 we introduce the planar HMM that extends the HMM concept to model images. The planar segmentation problem for PHMM is defined in section 4. It was recently shown (Kearns and Levin, 1992) that the planar segmentation problem is NP-hard, and therefore, in order to obtain an effective planar segmentation algorithm, we propose to constrain the parameters of the PHMM. We show sufficient conditions in terms of PHMM parameters for such algorithm to exist and describe the algorithm. This approach differs from the one taken in references (Chellappa and Chatterjee, 1985) and (Derin and Elliot, 1987), where instead of restricting the problem, a suboptimal solution to the general problem was found. Since in (Kearns and Levin, 1992) it was also shown that planar segmentation problem is hard to approximate, such suboptimal solution doesn't have any guaranteed bounds. The segmentation algorithm can now be used effectively not only for aligning isolated images, but also for joint recognition/segmentation, eliminating the need of independent segmentation that usually leads to unrecoverable errors in recognition. The same algorithm is used for estimation of the parameters of the model given a set of example images. In section 5, results of isolated hand-written digit recognition experiments are presented. The results indicate that even in the simple case of isolated characters, the elimination of planar distortions enhances the performance significantly. Section 6 contains the summary of this work.

## 2   Hidden Markov Model

The HMM is a statistical model that is used to describe temporal signals $G = \{g(t) : 1 \leq t \leq T, g \in \mathbf{G} \subset \mathbf{R}^n\}$ in speech processing applications (Rabiner, 1989; Lee et al., 1990; Wilpon et al., 1990; Pieraccini and Levin, 1991). The HMM is a composite statistical source comprising a set $\mathbf{s} = \{1, \cdots, T_R\}$ of $T_R$ sources called states. The $i$-th state, $i \in \mathbf{s}$, is characterized by its probability distribution $P_i(g)$ over $\mathbf{G}$. At each time $t$ only one of the states is active, emitting the observable $g(t)$. We denote by $s(t)$, $s(t) \in \mathbf{s}$ the random variable corresponding to the active state at time $t$. The joint probability distribution (for real-valued $g$) or discrete probability mass (for $g$ being a discrete variable) $P(s(t), g(t))$ for $t > 1$ is characterized by the following property:

$$P(s(t), g(t) \mid s(1{:}t{-}1), g(1{:}t{-}1)) = P(s(t) \mid s(t{-}1)) P(g(t) \mid s(t)) \equiv \qquad (1)$$

$$= P(s(t) \mid s(t{-}1)) P_{s(t)}(g(t)),$$

where $s(1{:}t{-}1)$ stands for the sequence $\{s(1), \cdots s(t{-}1)\}$, and $g(1{:}t{-}1) = \{g(1), \ldots, g(t{-}1)\}$. We denote by $a_{ij}$ the transition probability $P(s(t){=}j \mid s(t{-}1){=}i)$, and by $\pi_i$, the probability of state $i$ being active at $t{=}1$, $\pi_i = P(s(1){=}i)$. The probability of the entire sequence of states $S \equiv s(1{:}T)$ and observations $G = g(1{:}T)$ can be expressed as

$$P(G, S) = \pi_{s(1)} P_{s(1)}(g(1)) \prod_{t=2}^{T} a_{s(t-1)s(t)} \, P_{s(t)}(g(t)). \qquad (2)$$

The interpretation of equations (1) and (2) is that the observable sequence $G$ is generated in two stages: first, a sequence $S$ of $T$ states is chosen according to the Markovian distribution parametrized by $\{a_{ij}\}$ and $\{\pi_i\}$; then each one of the states $s(t)$, $1 \leq t \leq T$, in $S$ generates an observable $g(t)$ according to its own memoryless distribution $P_{s(t)}$, forming the observable sequence $G$. This model is called a *hidden* Markov model, because the state sequence $S$ is not given, and only the observation sequence $G$ is known. A particular case of this model, called a *left-to-right* HMM, where $a_{ij} = 0$ for $j < i$, and

$\pi_1 = 1$, is especially useful for speech recognition. In this case each state of the model represents an unspecified acoustic unit, and due to the "left-to-right" structure, the whole word is modeled as a concatenation of such acoustic units. The time spent in each of the states is not fixed, and therefore the model can take into account the duration variability between different utterances of the same word.

The segmentation problem of HMM is that of estimating the most probable state sequence $\hat{S}$, given the observation $G$,

$$\hat{S} = \underset{S}{argmax}\, P(S \mid G) = \underset{S}{argmax}\, P(G,S). \tag{3}$$

The problem of finding $\hat{S}$ through exhaustive search is of exponential complexity, since there exist $T^{T_R}$ possible state sequences, but it can be solved in polynomial time using a dynamic programming approach (i.e. Viterbi algorithm). The segmentation plays a central role in all HMM-based speech recognizers, since for connected speech it gives the segmentation into words or sub-word units, and performs a recognition simultaneously, in an optimal way. This is in contrast to sequential systems, in which the connected speech is first segmented into words/subwords according to some rules, and than the individual segments are recognized by computing the appropriate likelihoods, and where many recognition errors are caused by unrecoverable segmentation errors. Higher-level syntactical knowledge can be integrated into decoding process through transition probabilities between the models. The segmentation is also used for estimating the HMMs parameters using a corpus of a training data.

## 3    The Two-Dimensional Case: Planar HMM

In this section we describe a statistical model for planar image $G = \{g(x,y):(x,y) \in L_{X,Y},\, g \in \mathbf{G}\}$. We call this model "Planar HMM" (PHMM) and design it to extend the advantages of conventional HMM to the two-dimensional case.

The PHMM is a composite source, comprising a set $\mathbf{s} = \{(\tilde{x},\tilde{y}),\, 1 \leq \tilde{x} \leq X_R,\, 1 \leq \tilde{y} \leq Y_R\}$ of $N = X_R Y_R$ states. Each state in $\mathbf{s}$ is a stochastic source characterized by its probability density $P_{\tilde{x},\tilde{y}}(g)$ over the space of observations $g \in \mathbf{G}$. It is convenient to think of the states of the model as being located on a rectangular lattice where each state corresponds to a pixel of the corresponding reference image. Similarly to the conventional HMM, only one state is active in the generation of the $(x,y)$-th image pixel $g(x,y)$. We denote by $s(x,y) \in \mathbf{s}$ the active state of the model that generates $g(x,y)$. The joint distribution governing the choice of active states and image values has the following Markovian property:

$$P(g(x,y), s(x,y) \mid g(1{:}X, 1{:}y-1), g(1{:}x-1,y), s(1{:}X, 1{:}y-1), s(1{:}x-1,y)) = \tag{4}$$

$$= P(g(x,y) \mid s(x,y))\, P(s(x,y) \mid s(x-1,y), s(x,y-1)) =$$

$$= P_{s(x,y)}(g(x,y))\, P(s(x,y) \mid s(x-1,y), s(x,y-1)) =$$

where $g(1{:}X,y-1) \equiv \{g(x,y):(x,y) \in R_{X,y-1}\}$, $g(1{:}x-1,y) \equiv \{g(1,y), \cdots, g(x-1,y)\}$, and $s(1{:}X, 1{:}y-1)$, $s(1{:}x-1,y)$ are the active states involved in generating $g(1{:}X,y-1)$, $g(1{:}x-1,y)$, respectively, and $R_{X,y-1}$ is an axis parallel rectangle between the origin and the point $(X,y-1)$. Similarly to the one-dimensional case, it is useful to define a *left-to-right bottom-up* PHMM where $P(s(x,y)=(m,n) \mid s(x-1,y)=(i,j), s(x,y-1)=(k,l)) \neq 0$ only when $i \leq m$ and $l \leq n$, that does not allow for "fold overs" in the state image. The Markovian property (4) allows the left-to-right bottom-up PHMM to model elastic distortions among different realizations of the same image, similarly to the way the Markovian property in left-to-right HMM handles temporal alignment. We have chosen this definition (4) of Markovian property rather than others (see for example Derin and Kelly, 1989) since it leads to the formulation of a segmentation problem which is similar to the planar alignment defined in (Levin and Pieraccini, 1992).

Using property (4), the joint likelihood of the image $G = g(1{:}X, 1{:}Y)$ and the state image $S = s(1{:}X, 1{:}Y)$ can be written as

$$P(G,S) = \prod_{x=1}^{X} \prod_{y=1}^{Y} P_{s(x,y)}(g(x,y)) \tag{5}$$

$$\pi_{s(1,1)} \prod_{x=2}^{X} a^{H}_{s(x-1,1),s(x,1)} \prod_{y=2}^{Y} a^{V}_{s(1,y-1),s(1,y)} \prod_{y=2}^{Y} \prod_{x=2}^{X} A_{s(x-1,y),s(x,y-1),s(x,y)},$$

where:

$$A_{(i,j),(k,l),(m,n)} \equiv P(s(x,y)=(m,n) \mid s(x-1,y)=(i,j), s(x,y-1)=(k,l)),$$

$$a^{H}_{(i,j),(m,n)} \equiv P(s(x,1)=(m,n) \mid s(x-1,1)=(i,j)),$$

$$a^{V}_{(k,l),(m,n)} \equiv P(s(1,y)=(m,n) \mid s(1,y)=(k,l)),$$

and

$$\pi_{ij} \equiv P(s(1,1)=(i,j))$$

denote the generalized transition probabilities of PHMM. Similarly to HMM, (5) suggests that an image $G$ is generated by the PHMM in two successive stages: in the first stage the state matrix $S$ is generated according to the Markovian probability distribution parametrized by $\{A\}$, $\{a^{H}\}$, $\{a^{V}\}$, and $\{\pi\}$. In the second stage, the image value in the $(x,y)$-th pixel is produced independently from other pixels according to the distribution of the $s(x,y)$-th state $P_{s(x,y)}(g)$. As in HMM, the state matrix $S$ in most of the applications is not known, only $G$ is observed.

## 4   Planar Segmentation Problem

The segmentation problem of PHMM is that of finding the state matrix $\hat{S}$ that best explains the observable image $G$ and defines an optimal alignment of the image to the model. Solving this problem eliminates the sensitivity to intra-class elastic distortions and allows for simultaneous segmentation/recognition of images similarly to the one-dimensional case. $S$ can be estimated as in (3) by $S = \underset{S}{argmax}\, P(G,S)$. If we approach this maximization by exhaustive search, the computational complexity is exponential, since there are $(X_R Y_R)^{XY}$ different state matrices. Since the segmentation problem is NP-hard (Kearns and Levin, 1992), we suggest to simplify the problem by constraining the parameters of the PHMM, so that efficient segmentation algorithm can be found. In this section we present conditions in terms of the generalized transition probabilities of PHMM that are sufficient to guarantee that the most likely state image $\hat{S}$ can be computed in polynomial time, and describe an algorithm for doing that.

For the problem of finding $\hat{S}$ to be solved in polynomial time, there should exist a grouping of the set $s$ of states of the model into $N_G$ mutually exclusive[1] subsets of states $\gamma_p$, $s = \underset{p=1}{\overset{N_G}{U}} \gamma_p$. The generalized transition probabilities should satisfy the two following constraints with respect to such grouping:

$$A_{(i,j),(k,l),(m,n)} \neq 0 \;;\; a^{H}_{(i,j),(m,n)} \neq 0 \tag{6}$$

only if there exists $p$, $1 \leq p \leq N_G$, such that $(i,j), (m,n) \in \gamma_p$.

$$A_{(i,j),(k,l),(m,n)} = A_{(i,j),(k_1,l_1),(m,n)} \;;\; a^{V}_{(k,l),(m,n)} = a^{V}_{(k_1,l_1),(m,n)} \tag{7}$$

if there exists $p$, $1 \leq p \leq N_G$, such that $(k,l)$, $(k_1,l_1) \in \gamma_p$.

Condition (6) means that the the left neighbor $(i,j)$ of the state $(m,n)$ in the state matrix $S$ must be a member of the same subset $\gamma_p$ as $(m,n)$. Condition (7) means that the value of transition probability $A_{(i,j),(k,l),(m,n)}$ does not depend explicitly on the identity $(k,l)$ of the bottom neighboring state, but only on the subset $\gamma_p$ to which $(k,l)$ belongs.

Under (6) and (7) the most likely state matrix $\hat{S}$ can be found using an algorithm described in (Levin and Pieraccini, 1992). This algorithm makes use of the Viterbi procedure at two different levels. In the first stage optimal segmentation is computed for each subset $\gamma_p$ with each image raw using Viterbi. Then global segmentation is found, through Viterbi, by combining optimally the segmentations obtained in the previous stage.

Although conditions (6),(7) are hard to check in practice since any possible grouping of the states has to be considered, they can be effectively used in constructive mode, i.e., chosing one particular grouping, and then imposing the constraints (6) and (7) on the generalized transition probabilities with respect to this grouping. For example, if we choose $\gamma_p = \{ (\tilde{x}, \tilde{y}) \mid 1 \leq \tilde{x} \leq X_R, \tilde{y} = p \}$, $1 \leq p \leq Y_R$, then the constraints (6),(7) become:

$$A_{(i,j),(k,l),(m,n)} \neq 0, \; \overset{H}{a}_{(i,j),(m,n)} \neq 0 \text{ only for } j = n , \tag{8}$$

and,

$$A_{(i,j),(k,l),(m,n)} = A_{(i,j),(k_1,l),(m,n)}, \; \overset{V}{a}_{(k,l),(m,n)} = \overset{V}{a}_{(k_1,l),(m,n)} \text{ for } 1 \leq k_1, k \leq X_R. \tag{9}$$

Note that constraints (6), (7) break the symmetry between the roles of the two coordinates. Other sets of conditions can be obtained from (6) and (7) by coordinate transformation. For example, the roles of the vertical and the horizontal axes can be exchanged. A grouping and constraints set chosen for a particular application should reflect the geometric properties of the images.

## 5  Experimental Results

The PHMM approach was tested on a writer-independent isolated handwritten digit recognition application. The data we used in our experiments was collected from 12 subjects (6 for training and 6 for test). Each subject was asked to write 10 samples of each digit. Samples were written in fixed-size boxes, therefore naturally size-normalized and centered. Each sample in the database was represented by a $16 \times 16$ binary image.

Each character class (digit) was represented by a single PHMM, satisfying (6) and (7). Each PHMM had a *strictly* left-to-right bottom-up structure, where the state matrix $S$ was restricted to contain every state of the model, i.e., states could not be skipped. All models had the same number of states. Each state was represented by its own binary probability distribution, i.e., the probability of a pixel being 1 (black) or 0 (white). We estimated these probabilities from the training data with the following generalization of the Viterbi training algorithm (Jelinek, 1976). For the initialization we uniformly divided each training image into regions corresponding to the states of its model. The initial value of $P_i(g=1)$ for the $i$-th state was obtained as a frequency count of the black pixels in the corresponding region over all the samples of the same digit. Each iteration of the algorithm consisted of two stages: first, the samples were aligned with the corresponding model, by finding the best state matrix $S$. Then, a new frequency count for each state was used to update $P_i(1)$, according to the obtained alignment. We noticed that the training procedure converged usually after 2-4 iterations, and in all the experiments the algorithm was stopped at the 10th iteration. The recognition was performed by assigning the test sample to the class k for which the alignment likelihood was maximal.

Table 1 shows the number of errors in the recognition of the training set and the test set for different sizes of the models.

| Number of states | Recognition Errors | |
| :---: | :---: | :---: |
| $X_R = Y_R$ | Training | Test |
| 6 | 78 | 82 |
| 8 | 36 | 50 |
| 9 | 35 | 48 |
| 10 | 26 | 32 |
| 11 | 21 | 38 |
| 12 | 18 | 42 |
| 16 | 36 | 64 |

**Table 1:** Number of errors in the recognition of the training set and the test set for different size of the models (out of 600 trials in both cases)

It is worth noting the following two points. First, the test error shows a minimum for $X_R = Y_R = 10$ of 5%. By increasing or decreasing the number of states this error increases. This phenomenon is due to the following:

1.  The typical under/over parametrization behavior.

2.  Increasing the number of states closer to the size of the modeled images reduces the flexibility of the alignment procedure, making this a trivial uniform alignment when $X_R = Y_R = 16$.

Also, the training error decreases monotonically with increasing number of states up to $X_R = Y_R = 16$. This is again typical behavior for such systems, since by increasing the number of states, the number of model parameters grows, improving the fit to the training data. But when the number of states equals the dimensions of the sample images, $X_R = Y_R = 16$, there is a sudden significant increase in the training error. This behavior is consistent with point (2) above.

Fig. 1 shows three sets of models with different numbers of states. The states of the models in this figure are represented by squares, where the grey level of the square encodes the probability $P(g=1)$. The (6×6) state models have a very coarse representation of the digits, because the number of states is so small. The (10×10) state models appear much sharper than the (16×16) state models, due to their ability to align the training samples.

This preliminary experiment shows that eliminating elastic distortions by the alignment procedure discussed above plays an important role in the task of isolated character recognition, improving the recognition accuracy significantly. Note that the simplicity of this task does not stress the full power of the PHMM representation, since the data was isolated, size-normalized, and centered. On this task, the achieved performance is comparable to that of many other OCR systems. We expect that in harder tasks, involving connected text, the advantage of the proposed method will enhance the performance. Recently, this approach is being successfully applied to the task of recognition of noisy degraded printed messages (Agazzi et al., 1993).

## 6   Summary and Discussion

In this paper we describe a planar hidden Markov model and develop a planar segmentation algorithm that generalizes the Viterbi procedure widely used in speech recognition. This algorithm can be used to perform joint optimal recognition/segmentation of images incorporating some grammatical constraints and tolerating intra-class elastic distortions. The PHMM approach was tested on an isolated, hand-written digit recognition application. An analysis of the results indicate that even in a simple case of isolated characters, the elimination of elastic distortions enhances

recognition performance significantly. We expect that the advantage of this approach will be even more valuable in harder tasks, such as cursive writing recognition/spotting, for which an effective solution using the current available techniques has not yet been found.

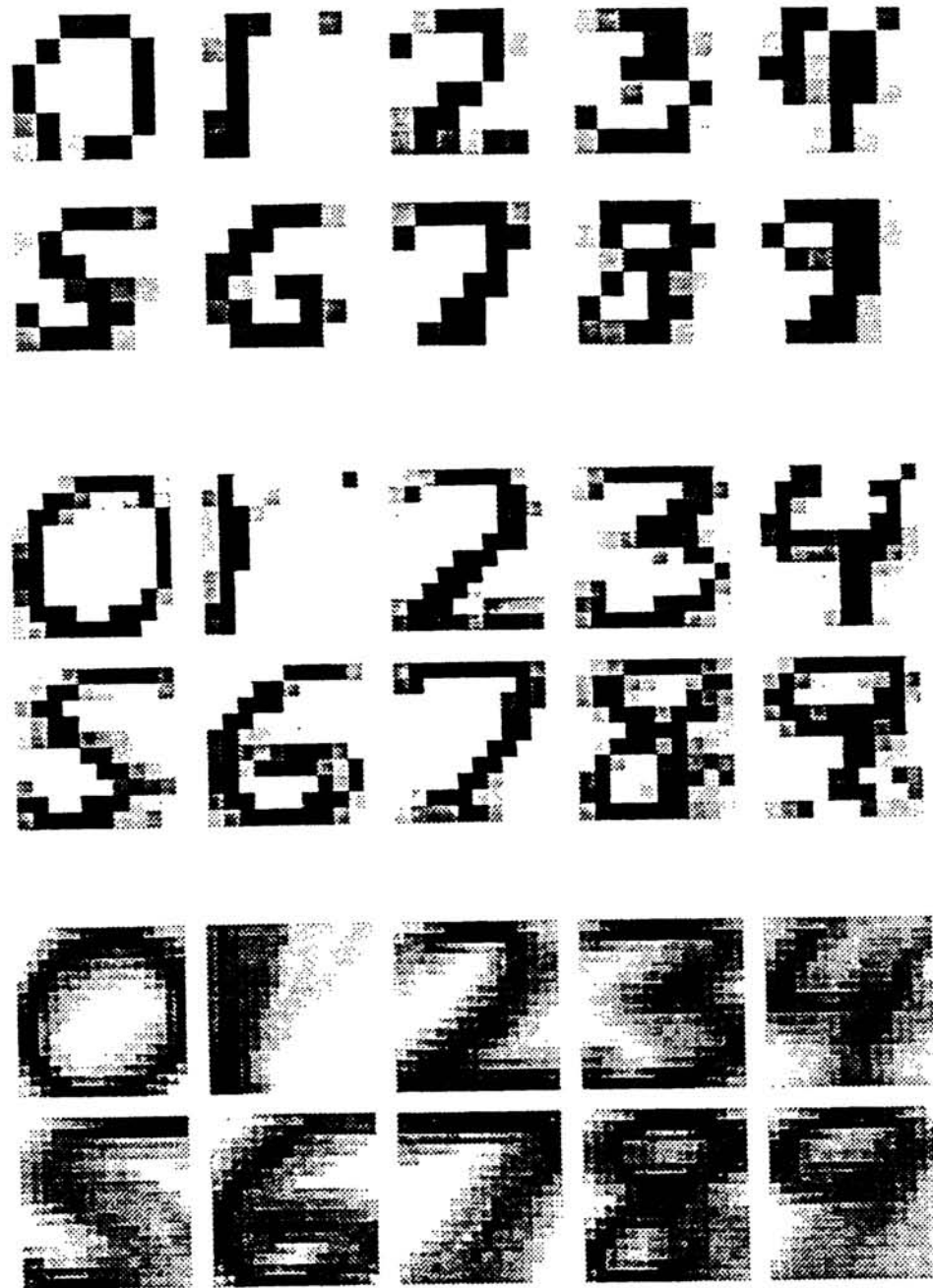

Figure 1: Three sets of models with 6×6, 10×10, and 16×16 states.

## Footnotes

[1] It is possible to drop the mutually exclusiveness constraints by duplicating states, but then we have to ensure that the number of subsets, $N_G$, should be polynomial in the dimensions of the model $X_R$, $Y_R$.

## References

O. E. Agazzi, S. S. Kuo, E. Levin, R. Pieraccini, " Connected and Degraded Text Recognition Using Planar Hidden Markov Models,"
*Proc. Of Int. COnference on Acoustics Speech and Signal Processing*, April 1993.

R. Chellappa, S. Chatterjee, "Classification of textures Using Gaussian Markov Random Fields," *IEEE Transactions on ASSP* , Vol. 33, No. 4, pp. 959-963, August 1985.

H. Derin, H. Elliot, "Modeling and Segmentation of Noisy and Textured Images Using Gibbs Random Fields," *IEEE Transactions on PAMI* , Vol. 9, No. 1 pp. 39-55, January 1987.

H. Derin, P. A. Kelly, 'Discrete-Index Markov-Type Random Processes,' in IEEE Proceedings, vol 77, #10, pp.1485-1510, 1989

G.D. Forney, "The Viterbi algorithm," *Proc. IEEE,* Mar. 1973.

F. Jelinek, "Continuous Speech Recognition by Statistical Methods," *Proceedings of IEEE,* vol. 64, pp. 532-556, April 1976.

M. Kearns, E. Levin, *Unpublished,* 1992.

C.-H. Lee, L. R. Rabiner, R. Pieraccini, J. G. Wilpon, "Acoustic Modeling for Large Vocabulary Speech Recognition," *Computer Speech and Language,* 1990, No. 4, pp. 127-165.

E. Levin, R. Pieraccini, "Dynamic Planar Warping and Planar Hidden Markov Modeling: from Speech to Optical Character Recognition," submitted to *IEEE Trans. on PAMI,* 1992.

R. Pieraccini, E. Levin, "Stochastic Representation of Semantic Structure for Speech Understanding," *Proceedings of EUROSPEECH 91,* Vol.2, pp. 383-386, Genova, September 1991.

L.R. Rabiner, "A Tutorial on Hidden Markov Models and Selected Applications in Speech Recognition," *Proc. IEEE,* Feb. 1989.

J. G. Wilpon, L. R. Rabiner, C.-H. Lee, E. R. Goldman, "Automatic Recognition of Keywords in Unconstrained Speech Using Hidden Markov Models," *IEEE Trans. on ASSP,* Vol. 38, No. 11, pp 1870-1878, November 1990.
